# Bias-Corrected Bootstrap and Model Uncertainty

**Harald Steck**[*]
MIT CSAIL
200 Technology Square
Cambridge, MA 02139
*harald@ai.mit.edu*

**Tommi S. Jaakkola**
MIT CSAIL
200 Technology Square
Cambridge, MA 02139
*tommi@ai.mit.edu*

## Abstract

The bootstrap has become a popular method for exploring model (structure) uncertainty. Our experiments with artificial and real-world data demonstrate that the graphs learned from bootstrap samples can be *severely biased towards too complex graphical models*. Accounting for this bias is hence essential, e.g., when exploring model uncertainty. We find that this bias is intimately tied to (well-known) spurious dependences induced by the bootstrap. The leading-order bias-correction equals *one half* of Akaike's penalty for model complexity. We demonstrate the effect of this simple bias-correction in our experiments. We also relate this bias to the bias of the plug-in estimator for entropy, as well as to the difference between the expected test and training errors of a graphical model, which asymptotically equals Akaike's penalty (rather than one half).

## 1 Introduction

Efron's bootstrap is a powerful tool for estimating various properties of a given statistic, most commonly its bias and variance (cf. [5]). It quickly gained popularity also in the context of model selection. When learning the structure of graphical models from small data sets, like gene-expression data, it has been applied to explore model (structure) uncertainty [7, 6, 8, 12].

However, the bootstrap procedure also involves various problems (e.g., cf. [4] for an overview). For instance, in the non-parametric bootstrap, where bootstrap samples $D^{(b)}$ ($b = 1, ..., B$) are generated by drawing the data points from the given data $D$ *with replacement*, each bootstrap sample $D^{(b)}$ often contains multiple *identical* data points, which is a typical property of *discrete* data. When the given data $D$ is in fact continuous (with a vanishing probability of two data points being identical), e.g., as in gene-expression data, the bootstrap procedure *introduces* a spurious discreteness in the samples $D^{(b)}$. A statistic computed from these *discrete* bootstrap samples may differ from the ones based on the *continuous* data $D$. As noted in [4], however, the effects due to this induced spurious discreteness are typically negligible.

In this paper, we focus on the *spurious dependences* induced by the bootstrap procedure, even when given discrete data. We demonstrate that the consequences of those

---

[*]Now at: ETH Zurich, Institute for Computational Science, 8092 Zurich, Switzerland.

spurious dependences cannot be neglected when exploring model (structure) uncertainty by means of bootstrap, whether parametric or non-parametric. Graphical models *learned from the bootstrap samples* are biased towards too complex models *and* this bias can be considerably larger than the variability of the graph structure, especially in the interesting case of limited data. As a result, too many edges are present in the learned model structures, and the confidence in the presence of edges is overestimated. This suggests that a *bias-corrected* bootstrap procedure is essential for exploring model structure uncertainty. Similarly to the statistics literature, we give a derivation for the bias-correction term to *amend* several popular scoring functions *when applied to bootstrap samples* (cf. Section 3.2). This bias-correction term asymptotically equals *one half* of the penalty term for model complexity in the Akaike Information Criterion (AIC), cf. Section 3.2. The (huge) effects of this bias and the proposed bias-correction are illustrated in our experiments in Section 5.

As the maximum likelihood score and the entropy are intimately tied to each other in the exponential family of probability distributions, we also relate this bias towards too complex models with the bias of the plug-in estimator for entropy (Section 3.1). Moreover, we show in Section 4, similarly to [13, 1], how the (bootstrap) bias-correction can be used to obtain a scoring function whose penalty for model complexity asymptotically equals Akaike's penalty (rather than one half of that).

## 2 Bootstrap Bias-Estimation and Bias-Correction

In this section, we introduce relevant notation and briefly review the bootstrap bias estimation of an arbitrary statistic as well as the bootstrap bias-correction (cf. also [5, 4]). The scoring-functions commonly used for graphical models such as the Akaike Information Criterion (AIC), the Bayesian Information Criterion (BIC), the Minimum Description Length (MDL), or the posterior probability, can be viewed as special cases of a statistic.

In a domain of $n$ discrete random variables, $X = (X_1, ..., X_n)$, let $p(X)$ denote the (unknown) *true* distribution from which the given data $D$ has been sampled. The *empirical* distribution implied by $D$ is then given by $\hat{p}(X)$, where $\hat{p}(x) = N(x)/N$, where $N(x)$ is the frequency of state $X = x$ and $N = \sum_x N(x)$ is the sample size of $D$. A statistic $T$ is any number that can be computed from the given data $D$. Its bias is defined as $\mathrm{Bias}_T = \langle T(D) \rangle_{D \sim p} - T(p)$, where $\langle T(D) \rangle_{D \sim p}$ denotes the expectation over the data sets $D$ of size $N$ sampled from the (unknown) true distribution $p$. While $T(D)$ is an arbitrary statistic, $T(p)$ is the associated, but possibly slightly different, statistic that can be computed from a (normalized) *distribution*. Since the *true* distribution $p$ is typically unknown, $\mathrm{Bias}_T$ cannot be computed. However, it can be approximated by the bootstrap bias-estimate, where $p$ is replaced by the empirical distribution $\hat{p}$, and the average over the data sets $D$ is replaced by the one over the bootstrap samples $D^{(b)}$ generated from $\hat{p}$, where $b = 1, ..., B$ with sufficiently large $B$ (e.g., cf. [5]):

$$\widehat{\mathrm{Bias}}_T = \langle T(D^{(b)}) \rangle_b - T(\hat{p}) \tag{1}$$

The estimator $T(\hat{p})$ is a so-called *plug-in* statistic, as the *empirical* distribution is "plugged in" in place of the (unknown) *true* one. For example, $\tilde{T}_{\sigma^2}(\hat{p}) = \mathbb{E}(X^2) - \mathbb{E}(X)^2$ is the familiar plug-in statistic for the variance, while $T_{\sigma^2}^{\mathrm{unbiased}}(D) = N/(N-1)T_{\sigma^2}(\hat{p})$ is the unbiased estimator.

Obviously, a *plug-in* statistic yields an unbiased estimate concerning the distribution that is plugged in. Consequently, when the *empirical* distribution is plugged in, a plug-in statistic typically does *not* give an unbiased estimate concerning the (unknown) *true* distribution. Only plug-in statistics that are *linear* functions of $\hat{p}(x)$ are inherently unbiased (e.g., the arithmetic mean). However, most statistics,

including the above scoring functions, are *non-linear* functions of $\hat{p}(x)$ (or equivalently of $N(x)$). In this case, the bias does *not* vanish in general. In the special case where a plug-in statistic is a convex (concave) function of $\hat{p}$, it follows immediately from the Jensen inequality that its bias is positive (negative). For example, the statistic $T_{\sigma^2}(\hat{p})$ is a negative quadratic, and thus concave, function of $\hat{p}$, and hence underestimates the variance of the (unknown) *true* distribution.

The general procedure of bias-correction can be used to reduce the bias of a biased statistic considerably. The bootstrap bias-corrected estimator $T^{\mathrm{BC}}$ is given by

$$T^{\mathrm{BC}}(D) = T(D) - \widehat{\mathrm{Bias}}_T = 2\,T(D) - \langle T(D^{(b)})\rangle_b, \qquad (2)$$

where $\widehat{\mathrm{Bias}}_T$ is the bootstrap bias estimate according to Eq. 1.[1] Typically, $T^{\mathrm{BC}}(D)$ agrees with the corresponding unbiased estimator in leading order in $N$ (cf., e.g., [5]). Higher-order corrections can be achieved by "bootstrapping the bootstrap" [5].

Bias-correction can be dangerous in practice (cf. [5]): even though $T^{\mathrm{BC}}(D)$ is less biased than $T(D)$, the bias-corrected estimator may have substantially larger variance. This is due to a possibly higher variability in the estimate of the bias, particularly when computed from small data sets. However, this is not an issue in this paper, since the "estimate" of the bias turns out to be independent of the empirical distribution (in leading order in $N$).

## 3  Bias-Corrected Scoring-Functions

In this section, we show that the above popular scoring-functions are (considerably) biased towards too complex models *when applied to bootstrap samples* (in place of the given data). These scoring functions can be amended by an *additional penalty term* that accounts for this bias. Using the bootstrap bias-correction in a slightly non-standard way, a simple expression for this penalty term follows easily (Section 3.2) from the well-know bias of the plug-in estimator of the entropy, which is reviewed in Section 3.1 (cf. also, e.g., [11, 2, 16]).

### 3.1  Bias-Corrected Estimator for True Entropy

The entropy of the (true) distribution $p(X)$ is defined by $H(p(X)) = -\sum_x p(x)\log p(x)$. Since this is a concave function of the $p$'s, the plug-in estimator $H(\hat{p}(X))$ tends to underestimate the true entropy $H(p(X))$ (cf. Section 2). The bootstrap bias estimate of $H(\hat{p}(X))$ is $\widehat{\mathrm{Bias}}_H = \langle H(D^{(b)})\rangle_b - H(\hat{p})$, where

$$\langle H(D^{(b)})\rangle_b = \frac{1}{B}\sum_{b=1}^{B} H(D^{(b)}(X)) = -\sum_x \langle \frac{\nu(x)}{N}\log\frac{\nu(x)}{N}\rangle_{\nu(x)\sim\mathrm{Bin}(N,\hat{p}(x))}, \qquad (3)$$

where $\mathrm{Bin}(N,\hat{p}(x))$ denotes the Binomial distribution that originates from the re-sampling procedure in the bootstrap; $N$ is the sample size; $\hat{p}(x)$ is the probability of sampling a data point with $X = x$. An exact evaluation of Eq. 3 is computationally prohibitive in most cases. Monte Carlo methods, while yielding accurate results, are computationally costly. An analytical approximation of Eq. 3 follows immediately from the second-order Taylor expansion of $L(q(x)) := q(x)\log q(x)$ about $\hat{p}(x)$, where $q(x) = \nu(x)/N$:[2]

$$-\sum_x \langle L(\frac{\nu(x)}{N})\rangle_{\nu(x)} = H(\hat{p}(x)) - \frac{1}{2}\sum_x L''(\hat{p}(x))\,\langle[\frac{\nu(x)}{N} - \hat{p}(x)]^2\rangle_{\nu(x)} + \mathcal{O}(\frac{1}{N^2})$$

$$= H(\hat{p}(x)) - \frac{1}{2N}(|X| - 1) + \mathcal{O}(\frac{1}{N^2}), \qquad (4)$$

where $-L''(\hat{p}(x)) = -1/\hat{p}(x)$ is the observed Fisher information evaluated at the empirical value $\hat{p}(x)$, and $\langle[\nu(x) - N\hat{p}(x)]^2\rangle_{\nu(x)} = N\hat{p}(x)(1 - \hat{p}(x))$ is the well-known variance of the Binomial distribution, induced by the bootstrap. In Eq. 4, $|X|$ is the number of (joint) states of $X$. The bootstrap bias-corrected estimator for the entropy of the (unknown true) distribution is thus given by $H^{\mathrm{BC}}(\hat{p}(X)) = H(\hat{p}(X)) + \frac{1}{2N}(|X| - 1) + \mathcal{O}(\frac{1}{N^2})$.

## 3.2 Bias-Correction for Bootstrapped Scoring-Functions

This section is concerned with the bias of popular scoring functions that is *induced by the bootstrap procedure*. For the moment, let us focus on the BIC when learning a Bayesian network structure $m$,

$$T_{\mathrm{BIC}}(D, m) = N \sum_{i=1}^{n} \sum_{x_i, \pi_i} \hat{p}(x_i, \pi_i) \log \frac{\hat{p}(x_i, \pi_i)}{\hat{p}(\pi_i)} - \frac{1}{2} \log N \cdot |\theta|. \quad (5)$$

The maximum likelihood involves a summation over all the variables ($i = 1, ..., n$) and all the joint states of each variable $X_i$ and its parents $\Pi_i$ according to graph $m$. The number of independent parameters in the Bayesian network is given by

$$|\theta| = \sum_{i=1}^{n} (|X_i| - 1) \cdot |\Pi_i| \quad (6)$$

where $|X_i|$ denotes the number of states of variable $X_i$, and $|\Pi_i|$ the number of (joint) states of its parents $\Pi_i$. Like other scoring-functions, the BIC is obviously intended to be applied to the *given* data. If done so, optimizing the BIC yields an "unbiased" estimate of the *true* network structure underlying the given data. However, when the BIC is applied to a *bootstrap sample* $D^{(b)}$(instead of the *given* data $D$), the BIC cannot be expected to yield an "unbiased" estimate of the *true* graph. This is because the maximum likelihood term in the BIC is biased when computed from the *bootstrap sample* $D^{(b)}$ instead of the *given* data $D$. This bias reads $\widehat{\mathrm{Bias}}_{T_{\mathrm{BIC}}} = \langle T_{\mathrm{BIC}}(D^{(b)})\rangle_b - T_{\mathrm{BIC}}(D)$. It differs conceptually from Eq. 1 in two ways. First, it is the (exact) bias *induced by the bootstrap procedure*, while Eq. 1 is a *bootstrap approximation* of the (unknown) true bias. Second, while Eq. 1 applies to a statistic in general, the last term in Eq. 1 necessarily has to be a plug-in statistic. In contrast, both terms involved in $\widehat{\mathrm{Bias}}_{T_{\mathrm{BIC}}}$ comprise the same general statistic.

Since the maximum likelihood term is intimately tied to the entropy in the exponential family of probability distributions, the leading-order approximation of the bias of the entropy carries over (cf. Eq. 4):

$$\widehat{\mathrm{Bias}}_{T_{\mathrm{BIC}}} = \frac{1}{2} \sum_{i=1}^{n} \left( \{|X_i| \cdot |\Pi_i| - 1\} - \{|\Pi_i| - 1\} \right) + \mathcal{O}(\frac{1}{N}) = \frac{1}{2}|\theta| + \mathcal{O}(\frac{1}{N}), \quad (7)$$

where $|\theta|$ is the number of independent parameters in the model, as given in Eq. 6 for Bayesian networks. Note that this bias is identical to *one half* of the penalty for model complexity in the Akaike Information Criterion (AIC). Hence, this bias due to the bootstrap cannot be neglected compared to the penalty terms inherent in all popular scoring functions. Also our experiments in Section 5 confirm the dominating effect of this bias when exploring model uncertainty.

This bias in the maximum likelihood gives rise to *spurious dependences induced by the bootstrap* (a well-known property). In this paper, we are mainly interested in structure learning of graphical models. In this context, the bootstrap procedure obviously gives rise to a (considerable) *bias towards too complex models*. As a consequence, too many edges are present in the learned graph structure, and the confidence in the presence of edges is overestimated. Moreover, the (undesirable) additional directed edges in Bayesian networks tend to point towards variables that already have a large number of parents. This is because the bias is proportional to

the number of joint states of the parents of a variable (cf. Eqs. 7 and 6). Hence, the amount of the induced bias generally *varies* among the different edges in the graph.

Consequently, the BIC has to be amended when applied to a bootstrap sample $D^{(b)}$ (instead of the given data $D$). The bias-corrected BIC reads $T_{\text{BIC}}^{BC}(D^{(b)}, m) = T_{\text{BIC}}(D^{(b)}, m) - \frac{1}{2}|\theta|$ (in leading order in $N$). Since the bias originates from the maximum likelihood term involved in the BIC, the same bias-correction applies to the AIC and MDL scores. Moreover, as the BIC approximates the (Bayesian) log marginal likelihood, $\log p(D|m)$, for large $N$, the leading-order bias-correction in Eq. 7 can also be expected to account for most of the bias of $\log p(D^{(b)}|m)$ when applied to bootstrap samples $D^{(b)}$.

## 4 Bias-Corrected Maximum-Likelihood

It may be surprising that the bias derived in Eq. 7 equals only *one half* of the AIC penalty. In this section, we demonstrate that this is indeed consistent with the AIC score. Using the standard bootstrap bias-correction procedure (cf. Section 2), we obtain a scoring function that asymptotically equals the AIC. This approach is similar to the ones in [1, 13].

Assume that we are given some data $D$ sampled from the (unknown) true distribution $p(X)$. The goal is to learn a Bayesian network model with $p(X|\hat{\theta}, m)$, or $\hat{p}(X|m)$ in short, where $m$ is the graph structure and $\hat{\theta}$ are the *maximum likelihood* parameter estimates, given data $D$. An information theoretic measure for the quality of graph $m$ is the KL divergence between the (unknown) true distribution $p(X)$ and the one described by the Bayesian network, $\hat{p}(X|m)$ (cf. the approach in [1]). Since the entropy of the true distribution $p(X)$ is an irrelevant constant when comparing different graphs, minimizing the KL-divergence is equivalent to minimizing the statistic

$$T(p, \hat{p}, m) = -\sum_x p(x) \log \hat{p}(x|m), \tag{8}$$

which is the *test error* of the learned model when using the log loss. When $p$ is unknown, one cannot evaluate $T(p, \hat{p}, m)$, but approximate it by the *training error*,

$$T(\hat{p}, m) = -\sum_x \hat{p}(x) \log \hat{p}(x|m) = -\sum_x \hat{p}(x|m) \log \hat{p}(x|m). \tag{9}$$

(assuming exponential family distributions). Note that $T(\hat{p}, m)$ is equal to the negative maximum log likelihood up to the irrelevant factor $N$. It is well-known that the training error underestimates the test error. However, the "bias-corrected training error",

$$T^{\text{BC}}(\hat{p}, m) = T(\hat{p}, m) - \text{Bias}_{T(\hat{p},m)}, \tag{10}$$

can serve as a surrogate, (nearly) unbiased estimator for the unknown test error, $T(p, \hat{p}, m)$, and hence as a scoring function for model selection. The bias is given by the difference between the expected training error and the expected test error,

$$\text{Bias}_T = \underbrace{\sum_x p(x|m) \langle \log \hat{p}(x|m) \rangle_{D\sim p}}_{=-H(p(X|m))-\frac{1}{2N}|\theta|+\mathcal{O}(\frac{1}{N^2})} - \underbrace{\sum_x \langle \hat{p}(x|m) \log \hat{p}(x|m) \rangle_{D\sim p}}_{=-H(p(X|m))+\frac{1}{2N}|\theta|+\mathcal{O}(\frac{1}{N^2})} \approx -\frac{1}{N}|\theta|. \tag{11}$$

The expectation is taken over the various data sets $D$ (of sample size $N$) sampled from the unknown true distribution $p$; $H(p(X|m))$ is the (unknown) conditional entropy of the true distribution. In the leading-order approximation in $N$ (cf. also Section 3.1), the number of independent parameters of the model, $|\theta|$, is given in Eq. 6 for Bayesian network. Note that both the expected test error and the expected training error give rise to *one half* of the AIC penalty each. The overall

bias amounts to $|\theta|/N$, which exactly equals the AIC penalty for model complexity. Note that, while the AIC asymptotically favors the same models as cross-validation [15], it typically does not select the *true* model underlying the given data, but a more complex model.

When the bootstrap estimate of the (exact) bias in Eq. 11 is inserted in the scoring function in Eq. 10, the resulting score may be viewed as the frequentist version of the (Bayesian) Deviance Information Criterion (DIC)[13] (up to a factor 2): while averaging over the distribution of the model parameters is natural in the Bayesian approach, this is mimicked by the bootstrap in the frequentist approach.

## 5    Experiments

In our experiments with artificial and real-world data, we demonstrate the crucial effect of the bias *induced by the bootstrap procedure*, when exploring model uncertainty. We also show that the penalty term in Eq. 7 can compensate for most of this (possibly large) bias in structure learning of Bayesian networks.

In the first experiment, we used data sampled from the alarm network (37 discrete variables, 46 edges). Comprising 300 and 1,000 data points, respectively, the generated data sets can be expected to entail some model structure uncertainty. We examined two different scoring functions, namely BIC and posterior probability (uniform prior over network structures, equivalent sample size $\alpha = 1$, cf. [10]). We used the K2 search strategy [3] because of its computational efficiency and its accuracy in structure learning, which is high compared to local search (even when combined with simulated annealing) [10]. This accuracy is due to the additional input required by the K2 algorithm, namely a correct topological ordering of the variables according to the true network structure. Consequently, the reported variability in the learned network structures tends to be smaller than the uncertainty determined by local search (without this additional information). However, we are mainly interested in the *bias* induced by the bootstrap here, which can be expected to be largely unaffected by the search strategy.

Although the true alarm network is known, we use the network structures learned from the *given* data $D$ as a reference in our experiments: as expected, the optimal graphs learned from our *small* data sets tend to be sparser than the original graph in order to avoid over-fitting (cf. Table 1).[3]

We generated 200 bootstrap samples from the given data $D$ (as suggested in [5]), and then learned the network structure from each. Table 1 shows that the bias *induced by the bootstrap procedure* is considerable for both the BIC and the posterior probability: it cannot be neglected compared to the standard deviation of the distribution over the number of edges. Also note that, despite the small data sets, the bootstrap yields graphs that have even more edges than the true alarm network. In contrast, Table 1 illustrates that this bias towards too complex models can be reduced dramatically by the bias-correction outlined in Section 3.2. However note that the bias-correction does not work perfectly as it is only the leading-order correction in $N$ (cf. Eq. 7).

The jackknife is an alternative resampling method, and can be viewed as an approximation to the bootstrap (e.g., cf. [5]). In the delete-$d$ jackknife procedure, subsamples are generated from the given data $D$ by deleting $d$ data points.[4] The choice $d = 1$ is most popular, but leads to inconsistencies for non-smooth statistics (e.g., cf. [5]). These inconsistency can be resolved by choosing a larger value for

| | alarm network data | | | | pheromone |
| | N = 300 | | N = 1,000 | | N = 320 |
| | BIC | posterior | BIC | posterior | posterior |
|---|---|---|---|---|---|
| data $D$ | 41 | 40 | 43 | 44 | $63.0 \pm 1.5$ |
| boot BC | $40.7 \pm 4.9$ | $40.5 \pm 3.5$ | $44.2 \pm 2.6$ | $44.1 \pm 2.9$ | $57.8 \pm 3.5$ |
| boot | $49.1 \pm 11.5$ | $47.8 \pm 10.9$ | $47.3 \pm 4.6$ | $47.9 \pm 4.8$ | $135.7 \pm 51.1$ |
| jack 1 | $41.0 \pm 0.0$ | $40.0 \pm 0.0$ | $43.0 \pm 0.0$ | $44.0 \pm 0.0$ | $63.2 \pm 1.5$ |
| jack $d$ | $41.1 \pm 0.9$ | $40.1 \pm 0.3$ | $43.1 \pm 0.3$ | $43.7 \pm 0.4$ | $63.1 \pm 2.3$ |

Table 1: Number of edges (mean ± standard deviation) in the network structures learned from the given data set $D$, and when using various resampling methods: bias-corrected bootstrap (boot BC), naive bootstrap (boot), delete-1 jackknife (jack 1), and delete-$d$ jackknife (jack $d$; here $d = N/10$).

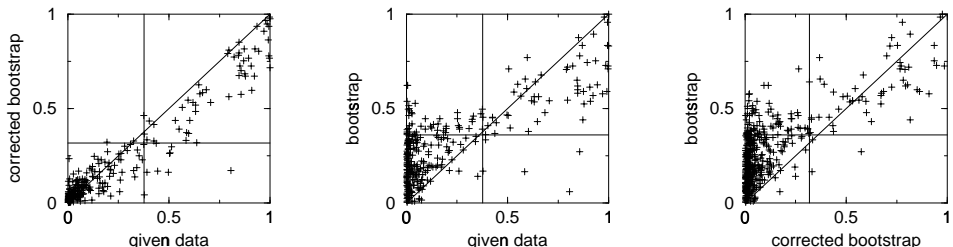

Figure 1: The axis of these scatter plots show the confidence in the presence of the edges in the graphs learned from the pheromone data. The vertical and horizontal lines indicate the threshold values according to the mean number of edges in the graphs determined by the three methods (cf. Table 1).

$d$, roughly speaking $\sqrt{N} < d \ll N$, cf. [5]. The underestimation of both the bias and the variance of a statistic is often considered a disadvantage of the jackknife procedure: the "raw" jackknife estimates of bias and variance typically have to be multiplied by a so-called "inflation factor", which is usually of the order of the sample size $N$. In the context of model selection, however, one may take advantage of the extremely small bias of the "raw" jackknife estimate when determining, e.g., the mean number of edges in the model. Table 1 shows that the "raw" jackknife is typically less biased than the bias-corrected bootstrap in our experiments. However, it is not clear in the context of model selection as to how meaningful the "raw" jackknife estimate of model variability is.

Our second experiment essentially confirms the above results. The yeast pheromone response data contains 33 variables and 320 data points (measurements) [9]. We discretized this gene-expression data using the average optimal number of discretization levels for each variable as determined in [14]. Unlike in [14], we simply discretized the data in a preprocessing step, and then conducted our experiments based on this discretized data set.[5] Since the correct network structure is unknown in this experiment, we used local search combined with simulated annealing in order to optimize the BIC score and the posterior probability ($\alpha = 25$, cf. [14]). As a reference in this experiment, we used 320 network structures learned from the *given* (discretized) data $D$, each of which is the highest-scoring graph found in a run of local search combined with simulated annealing.[6] Each resampling procedure is also based on 320 subsamples.

While the pheromone data experiments in Table 1 qualitatively confirm the previous results, the bias induced by the bootstrap is even larger here. We suspect that this difference in the bias is caused by the rather extreme parameter values in the original alarm network model, which leads to a relatively large signal-to-noise ratio even in small data sets. In contrast, gene-expression data is known to be extremely noisy.

Another effect of the spurious dependences induced by the bootstrap procedure is shown in Figure 1: the overestimation of the confidence in the presence of individual edges in the network structures. The confidence in an individual edge can be estimated as the ratio between the number of learned graphs where that edge is present and the overall number of learned graphs. Each mark in Figure 1 corresponds to an edge, and its coordinates reflect the confidence estimated by the different methods. Obviously, the naive application of the bootstrap leads to a considerable overestimation of the confidence in the presence of many edges in Figure 1, particularly of those whose *absence* is favored by both our reference and the bias-corrected bootstrap. In contrast, the confidence estimated by the bias-corrected bootstrap aligns quite well with the confidence determined by our reference in Figure 1, leading to more trustworthy results in our experiments.

## Footnotes

[1]Note that $\langle T(D^{(b)})\rangle_b$ is *not* the bias-corrected statistic.

[2]Note that this approximation can be applied analogously to $\mathrm{Bias}_H$ (instead of the *bootstrap estimate* $\widehat{\mathrm{Bias}}_H$), and the same leading-order term is obtained.

[3]Note that the greedy K2 algorithm yields exactly one graph from each given data set.

[4]As a consequence, unlike bootstrap samples, jackknife samples do not contain multiple identical data points when generated from a given continuous data set (cf. Section 1).

[5]Of course, the bias-correction according to Eq. 7 also applies to the joint optimization of the discretization and graph structure when given a bootstrap sample.

[6]Using the annealing parameters as suggested in [10], each run of simulated annealing resulted in a different network structure (local optimum) in practice.

# References

[1] H. Akaike. Information theory and an extension of the maximum likelihood principle. *International Symposium on Information Theory*, pp. 267–81. 1973.

[2] Carlton.On the bias of information estimates.*Psych. Bulletin*, 71:108–13, 1969.

[3] G. Cooper and E. Herskovits. A Bayesian method for constructing Bayesian belief networks from databases. *UAI*, pp. 86–94. 1991.

[4] A.C. Davison and D.V. Hinkley. *Bootstrap methods and their application*. 1997.

[5] B. Efron and R. J. Tibshirani. *An Introduction to the Bootstrap*. 1993.

[6] N. Friedman, M. Goldszmidt, and A. Wyner. Data analysis with Bayesian networks: A bootstrap approach. *UAI*, pp. 196–205. 1999.

[7] N. Friedman, M. Goldszmidt, and A. Wyner. On the application of the bootstrap for computing confidence measures on features of induced Bayesian networks. *AI & Stat.*, p.p 197–202. 1999.

[8] N. Friedman, M. Linial, I. Nachman, and D. Pe'er. Using Bayesian networks to analyze expression data. *Journal of Computational Biology*, 7:601–20, 2000.

[9] A. J. Hartemink, D. K. Gifford, T. S. Jaakkola, and R. A. Young. Combining location and expression data for principled discovery of genetic regulatory networks. In *Pacific Symposium on Biocomputing*, 2002.

[10] D. Heckerman, D. Geiger, and D. M. Chickering. Learning Bayesian networks: The combination of knowledge and statistical data. *Machine Learning*, 20:197–243, 1995.

[11] G. A. Miller. Note on the bias of information estimates. *Information Theory in Psychology: Problems and Methods*, pages 95–100, 1955.

[12] D. Pe'er, A. Regev, G. Elidan, and N. Friedman. Inferring subnetworks from perturbed expression profiles. *Bioinformatics*, 1:1–9, 2001.

[13] D. J. Spiegelhalter, N. G. Best, B. P. Carlin, and A. van der Linde. Bayesian measures of model complexity and fit. *J. R. Stat. Soc. B*, 64:583–639, 2002.

[14] H. Steck and T. S. Jaakkola. (Semi-)predictive discretization during model selection. *AI Memo 2003-002, MIT*, 2003.

[15] M. Stone. An asymptotic equivalence of choice of model by cross-validation and Akaike's criterion. *J. R. Stat. Soc. B*, 36:44–7, 1977.

[16] J. D. Victor. Asymptotic bias in information estimates and the exponential (Bell) polynomials. *Neural Computation*, 12:2797–804, 2000.
